# COMPUTER MODELING OF ASSOCIATIVE LEARNING

DANIEL L. ALKON[1]   FRANCIS QUEK[2a]   THOMAS P. VOGL[2b]

1. Laboratory for Cellular and Molecular
   Neurobiology, NINCDS, NIH, Bethesda, MD 20892

2. Environmental Research Institute of Michigan
   a) P.O. Box 8618, Ann Arbor, MI 48107
   b) 1501 Wilson Blvd., Suite 1105, Arlington, VA 22209

## INTRODUCTION

Most of the current neural networks use models which have only tenuous connections to the biological neural systems on which they purport to be based, and negligible input from the neuroscience/biophysics communities. This paper describes an ongoing effort which approaches neural net research in a program of close collaboration of neuroscientists and engineers. The effort is designed to elucidate associative learning in the marine snail Hermissenda crassicornis, in which Pavlovian conditioning has been observed. Learning has been isolated in the four neuron network at the convergence of the visual and vestibular pathways in this animal, and biophysical changes, specific to learning, have been observed in the membrane of the photoreceptor B cell. A basic charging capacitance model of a neuron is used and enhanced with biologically plausible mechanisms that are necessary to replicate the effect of learning at the cellular level. These mechanisms are non-linear and are, primarily, instances of second order control systems (e.g., fatigue, modulation of membrane resistance, time dependent rebound), but also include shunting and random background firing. The output of the model of the four-neuron network displays changes in the temporal variation of membrane potential similar to those observed in electrophysiological measurements.

## NEUROPHYSIOLOGICAL BACKGROUND

Alkon[1] showed that Hermissenda crassicornis, a marine snail, is capable of associating two stimuli in a fashion which exhibits all the characteristics of classical Pavlovian conditioning (acquisition, retention, extinction, and savings)[2]. In these experiments, Hermissenda were trained to associate a visual with a vestibular stimulus. In its normal environment, Hermissenda moves toward light; in turbulence, the animal increases the area of contact of its foot with the surface on which it is moving, reducing its forward velocity. Alkon showed that the snail can be condi-tioned to associate these stimuli through repeated exposures to ordered pairs (light followed by turbulence).

When the snails are exposed to light (the unconditioned stimulus) followed by turbulence (the conditioned stimulus) after varying time intervals, the snails transfer to the light their unconditioned response to turbulence (increased area of foot contact); i.e., when presented with light alone, they respond with an increased area of foot contact. The effect of such training lasts for several weeks. It was further shown that the learning was maximized when rotation followed light by a fixed interval of about one second, and that such learning exhibits all the characteristics of classical conditioning observed in higher animals.

The relevant neural interconnections of Hermissenda have been mapped by Alkon, and learning has been isolated in the four neuron sub-network (Figure 1) at the con-vergence of the visual and vestibular pathways of this animal. Light generates signals in the B cells while turbulence is transduced into signals by the statocyst's hair cells, the animal's vestibular organs. The optic ganglion cell mediates the interconnections between the two sensory pathways.

The effects of learning also have been observed at the cellular level. Alkon et al.[3] have shown that bio-physical changes associated with learning occur in the photo-receptor B cell of Hermissenda. The signals in the neurons take the form of voltage dependent ion

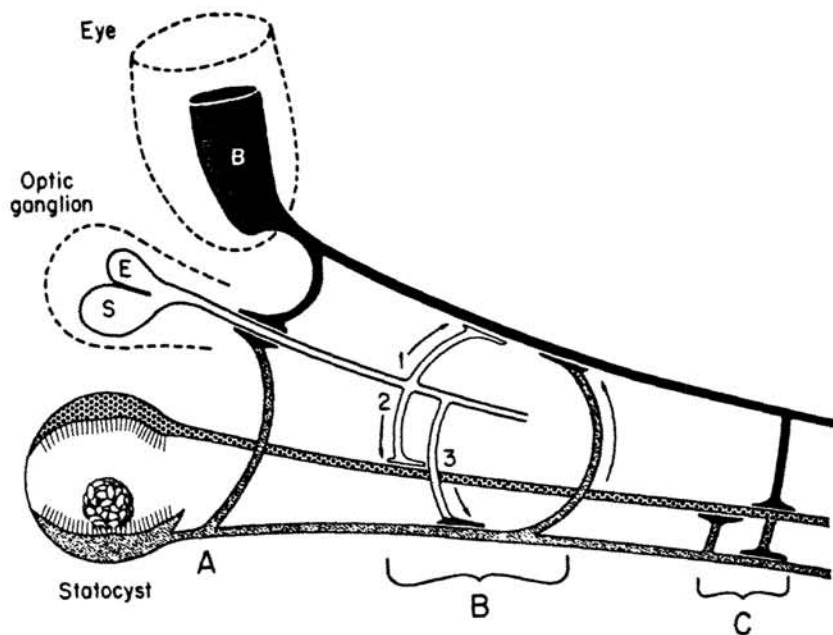

Figure 1. The four neuron network at the convergence of the visual and vestibular pathways of Hermissenda crassicornis. All filled endings indicate inhibitory synapses; all open endings indicate excitatory synapses.
(a) Convergence of synaptic inhibition from the photoreceptor B cell and caudal hair cells unto the optic ganglion cell.
(b) Positive synaptic feedback onto the type B photoreceptor: 1, direct synaptic excitation; 2, indirect excitation -- the ganglion excites the cephalic hair cell that inhibits the caudal hair cell, and thus disinhibits the type B cell; 3, indirect excitation -- the ganglion inhibits the caudal hair cell and thus disinhibits the type B cell.
(c) Intra- and intersensory inhibition. The cephalic and caudal hair cells are mutually inhibitory. The B cell inhibits mainly the cephalic hair cell.
From: Tabata, M., and Alkon, D. L. Positive synaptic feedback in visual system of nudibranch mollusk Hermissenda Crassicornis. J. of Neurophysiology 48: 174-191 (1982).

currents, and learning is reflected in biophysical changes in the membrane of the B cell. The effects of ion currents can be observed in the form of time variations in membrane potential recorded by means of microelectrodes. It is the variation in membrane potential resulting from associative learning that is the focus of this research.

Our goal is to model those properties of biological neurons sufficient (and necessary) to demonstrate associative learning at the neural network level. In order to understand the effect and necessity of each component of the model, a minimalist approach was adopted. Nothing was added to the model which was not necessary to produce a required effect, and then only when neurophysiologists, biophysicists, electrical engineers, and computer scientists agreed that the addition was reasonable from the perspective of their disciplines.

## METHOD

Following Kuffler and Nicholas[4], the model is described in terms of circuit elements. It must be emphasized however, that this is simply a recognition of the fact that the chemical and physical processes occurring in the neuron can be described by (partial) differential equations, as can electronic circuits. The equivalent circuit of the charging and discharging of the neuron membrane is shown in Figure 2. The model was constructed using the P3 network simulation shell developed by Zipser and Rabin[5]. The P3 strip-chart capability was particularly useful in facilitating interdisciplinary interactions. Figure 3 shows the response of the basic model of the neuron as the frequency of input pulses is varied.

Our aim, however, is not to model an individual neuron. Rather, we consistently focus on properties of the neural network that are necessary and sufficient to demonstrate associative learning. Examination of the behavior of biological neurons reveals additional common properties that express themselves differently depending on the function of the individual neuron. These properties include background firing, second order controls, and shunting. Their inclusion in the model is necessary for the simulation of associative learning, and their implementation is described below.

**INPUT CIRCUITRY**

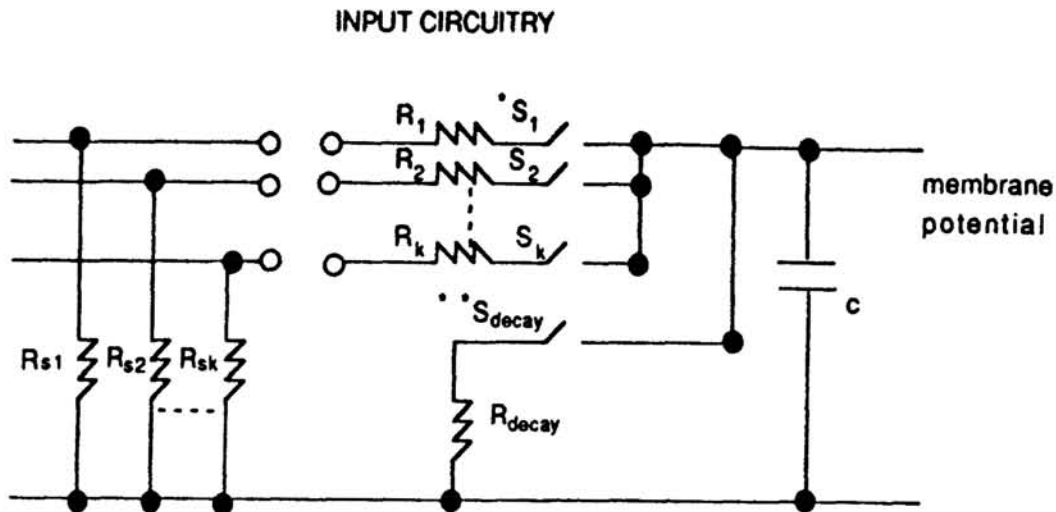

$\overset{\bullet}{S}_1 - S_k$    : closes when there are input EPSPs to the
            the cell, otherwise , open.

$\overset{\bullet \;\bullet}{S}_{decay}$    : closes when there is no input ESPSs to the
            the cell, otherwise, open.

Figure 2. Circuit model of the basic neuron. The lines
from the left are the inputs to the neuron from its
dendritic connections to presynaptic neurons. $R_{sk}$'s are
the resistances that determine the magnitude of the effect
(voltage) of the pulses from presynaptic neurons. The gap
indicated by open circles is a high impedance coupler. $R_1$
through $R_k$, together with C, determine the rise time for
the kth input of the potential across the capacitor C,
which represents the membrane. $R_{decay}$ controls the dis-
charge time constant of the capacitor C   When the
membrane potential (across C) exceeds threshold potential,
the neuron fires a pulse into its axon (to all output
connections) and the charge on C is reduced by the
"discharge quantum" (see text).

## BACKGROUND FIRING IN ALL NEURONS

Background firing, i.e., spontaneous firing by neurons
without any input pulses from other neurons, has been
observed in all neurons[6]. The fundamental importance of
background firing is exemplified by the fact that in the
four neuron network under study, the optic ganglion does

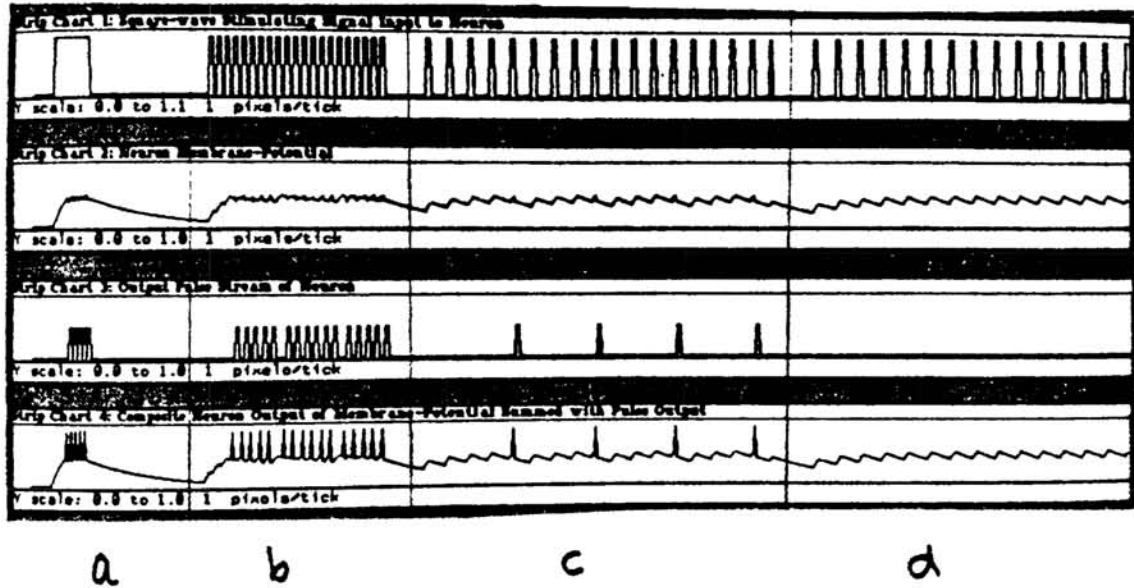

a          b               c                d

Figure 3.   Response of the basic model of a single neuron
to a variety of inputs.  The four horizontal strips, from
top to bottom, show: 1) the input stream; 2) the resulting
membrane potential; 3) the resulting stream of output
pulses; and 4) the composite output of pulses superimposed
on the membrane potential, emulating the corresponding
electrophysiological measurement.   The four vertical
sections, from left to right, indicate: a) an extended
input, simulating exposure of the B cell to light; b) a
presynaptic neuron firing at maximum frequency; c) a
presynaptic neuron firing at an intermediate frequency; d)
a presynaptic neuron firing at a frequency insufficient to
cause the neuron to fire but sufficient to maintain the
neuron at a membrane potential just below firing
threshold.

## BACKGROUND FIRING IN ALL NEURONS (Continued)

not have any synapses that excite it (all its inputs are
inhibitory).   However, the optic ganglion provides the
only two excitatory synapses in the entire network (one on
the photoreceptor B cell and the other on the cephalad
statocyst hair cell).  Hence, without back-ground firing,
i.e., when  there is  no external stimuli of the neurons,
all activity in the network would cease.

Further, without background firing, any stimulus to either
the vestibular or the visual receptors will completely
swamp the response of the network.

Background firing is incorporated in our model by applying random pulses to a 'virtual' excitatory synapse. By altering the mean frequency of the random pulses, various levels of 'internal' homeostatic neuronal activity can be simulated. Physiologically, this pulse source yields results similar to an ion pump or other energy source, e.g., cAMP, in the biological system.

## SECOND ORDER CONTROL IN NEURONS

Second order controls, i.e., the modulation of cellular parameters as the result of the past history of the neuron, appear in all biological neurons and play an essential role in their behavior. The ability of the cell to integrate its internal parameters (membrane potential in particular) over time turns out to be vital not only in understanding neural behavior but, more specifically, in providing the mechanisms that permit temporally specific associative learning. In the course of this investigation, a number of different second order control mechanisms, essential to the proper performance of the model, were elucidated. These mechanisms share a dependence on the time integral of the difference between the instantaneous membrane potential and some reference potential.

The particular second order control mechanisms incorporated into the model are: 1) Overshoot in the light response of the photoreceptor B cell; 2) maintenance of a post-stimulus state in the B cell subsequent to prolonged stimulation; 3) modulation of the discharge resistance of the B cell; 4) Fatigue in the statocysts and the optical ganglion; and 5) time dependent rebound in the optical ganglion. In addition to these second order control effects, the model required the inclusion of the observed shunting of competing inputs to the B cell during light exposure. The consequence of the interaction of these mechanisms with the basic model of the neurons in the four neuron network is the natural emergence of temporally specific associative learning.

## OVERSHOOT IN THE LIGHT RESPONSE OF THE PHOTORECEPTOR B CELL

Under strong light exposure, the membrane potential of an isolated photoreceptor B cell experiences an initial 'overshoot' and then settles at a rapidly firing level

far above the usual firing potential of the neuron (see Figure 4a). (We refer to the elevated membrane potential of the B cell during illumination as the "active firing membrane potential"). The initial overshoot (and slight ringing) observed in the potential of the biological B cell (Figure 4a) is the signature of an integral second order control system at work. This control was realized in the model by altering the quantity of charge removed from the cell (the discharge quantum) each time the cell fires. (The biological cell loses charge whenever it fires and the quantity of charge lost varies with the membrane potential.) The discharge quantum is modulated by the definite integral of the difference between the membrane potential and the active firing membrane potential as follows:

$$Q_{discharge}(t) = K \quad x \int_{t_o}^{t_1} \{Pot_{membrane}(t) - Pot_{active\ firing}(t)\}dt$$

As the membrane potential rises above the active firing membrane potential, the value of the integral rises. The magnitude of the discharge quantum rises with the integral. This increased discharge retards the membrane depolarization, until at some point, the size of the discharge quantum outstrips the charging effect of light on the membrane potential, and the potential falls. As the membrane potential falls below the active firing membrane potential, the magnitude of the discharge quantum begins to decrease (i.e., the value of the integral falls). This, in turn, causes the membrane potential to rise when the charging owing to light input once again overcomes the declining discharge quantum.

This process repeats with each subsequent swing in the membrane potential becoming smaller until steady state is reached at the active firing membrane potential. The response of the model to simulated light exposure is shown in Figure 4b. Note that only a single overshoot is obvious and that steady state is rapidly reached.

## MAINTAINING THE POST-STIMULUS STATE IN THE B CELL

During exposure to light, the B cell is strongly depolarized, and the membrane potential is maintained substantially above the firing threshold potential. When the light stimulus is removed, one would expect the cell to fire at its maximum rate so as to bring its membrane

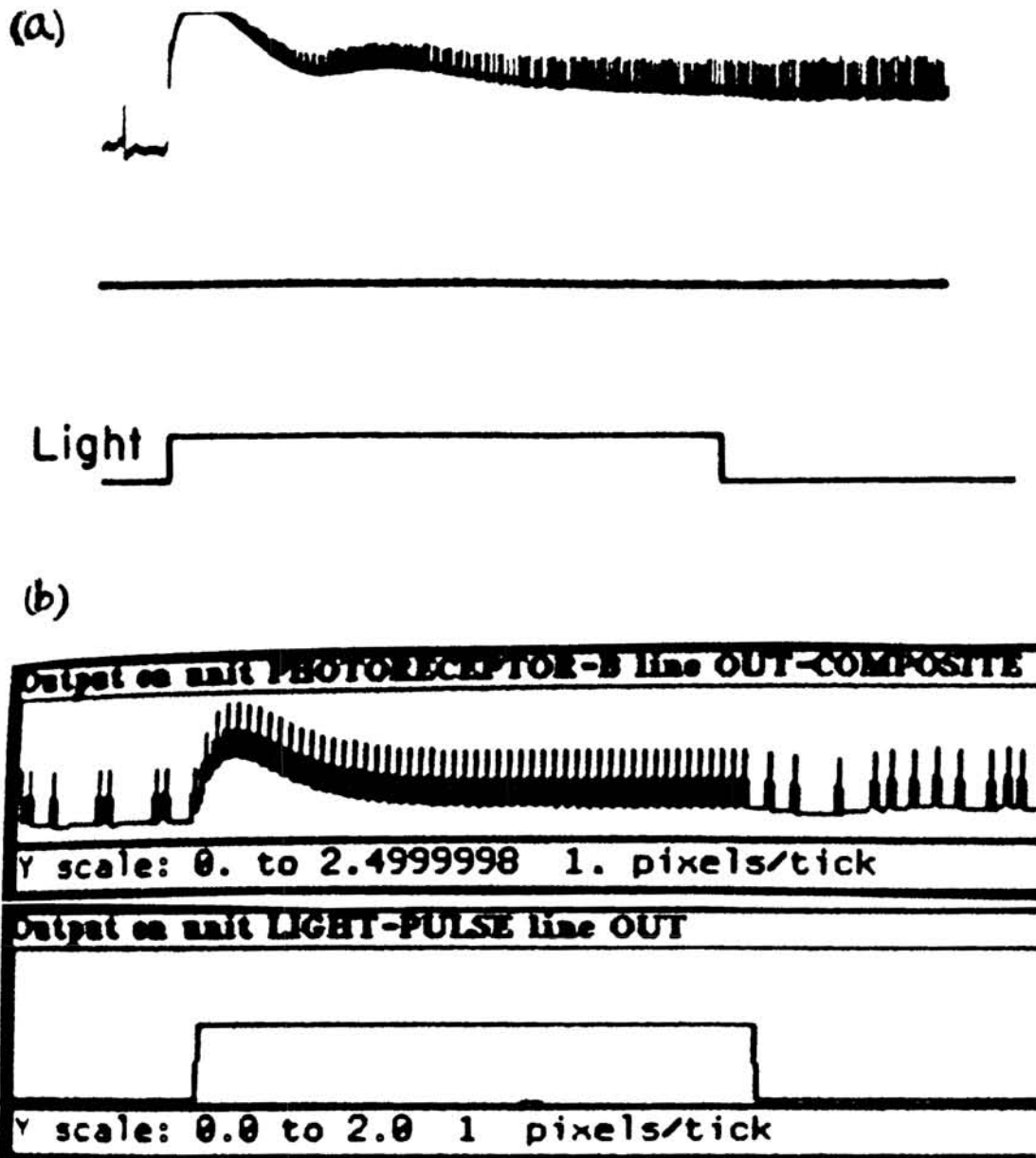

Figure 4.  Response of the B cell and the model to a light pulse.

(a). Electrophysiological recording of the response of the photoreceptor B cell to light.   Note the initial overshoot and one cycle of oscillation before the membrane potential settles at the "active firing potential." From: Alkon, D.L. Memory Traces in the Brain.    Cambridge University Press, London (1987), p.58.

(b) Response of the model to a light pulse.

potential below the firing threshold (by releasing a discharge quantum with each output pulse). This does not happen in Hermissenda; there is, however, a change in the amount of charge released with each output pulse when the cell is highly depolarized.

Note that the discharge quantum is modulated post-exposure in a manner analogous to that occurring during exposure as discussed above: It is modulated by the magnitude of the membrane potential above the firing threshold. The result of this modulation is that the more positive the membrane potential, the smaller the discharge quantum, (subject to a non-negative minimum value). The average value of the interval between pulses is also modulated by the magnitude of the discharge quantum. This modulation persists until the membrane potential returns to the firing threshold after cessation of light exposure.

This mechanism is particular to the B cell. Upon cessation of vestibular stimulation, hair cells fire rapidly until their membrane potentials are below the firing threshold, just as the basic model predicts.

## MODULATION OF DISCHARGE RESISTANCE IN THE B CELL

The duration of the post-stimulus membrane potential is determined by the magnitude of the discharge resistance of the B cell. In the model, the discharge resistance changes exponentially toward a predetermined maximum value, $R_{max}$, when the membrane potential exceeds the firing threshold. $R_{base}$ is the baseline value. That is,

$$R_{disch}(t-t_o) = R_{max} - \{R_{max} - R_{disch}(t_o)\}\exp\{(t-t_o)/\tau_{rise}\}$$

when the membrane potential is above the firing threshold, and

$$R_{disch}(t-t_o) = R_{base} - \{R_{base} - R_{disch}(t_o)\}\exp\{(t-t_o)/\tau_{decay}\}$$

when the membrane potential is below the firing threshold.

## FATIGUE IN STATOCYST HAIR CELLS

In Hermissenda, caudal cell activity actually decreases immediately after it has fired strongly, rather than returning to its normal background level of firing. This effect, which results from the tendency of membrane potential to "fatigue" toward its resting potential, is

incorporated into our model of the statocyst hair cells using the second order control mechanism previously described. I.e., when the membrane potential of a hair cell is above the firing threshold (e.g., during vestibular stimulation), the shunting resistance of the cell decays exponentially with time toward zero as long as the hair cell membrane potential is above the firing threshold. This resistance is allowed to recover exponentially to its usual value when the membrane potential falls below the firing threshold.

## FATIGUE OF THE OPTICAL GANGLION CELL DURING HYPER-POLARIZATION

In Hermissenda the optical ganglion undergoes hyperpolarization at the beginning of the light pulse and/or vestibular stimulus. Contrary to what one might expect, it then recovers and is close to the firing threshold by the time the stimuli cease. This effect is incorporated into the model by fatigue induced by hyperpolarization. As above, this fatigue is implemented by allowing the shunting resistance in the ganglion cell to decrease exponentially toward a minimum value, while the membrane potential is below the firing threshold by a prespecified amount. The value of the minimum shunting resistance is modulated by the magnitude of hyperpolarization (potential difference between the membrane potential and the firing threshold). The shunting resistance recovers exponentially from its hyperpolarized value, once the membrane potential returns to its firing threshold as a result of background firing input.

The effect of this decrease is that the ganglion cell will temporarily remain relatively insensitive to the enhanced post-stimulus firing of the B cell until the shunting resistance recovers. Once the membrane potential of the ganglion cell recovers, the pulses from the ganglion cell will excite the B cell and maintain its prolongation effect. (See Figure 1.)

The modulation of the minimum shunting resistance by the magnitude of hyperpolarization introduces the first stimulus pairing dependent component in the post-stimulus behavior of the B cell because the degree of hyperpolarization is higher under paired stimulus conditions.

## TIME DEPENDENT REBOUND IN THE OPTICAL GANGLION CELL

Experimental evidence with Hermissenda indicates that the rebound of the optical ganglion is much stronger than is possible if the usual background activity were the sole cause of this rebound. Furthermore rebound in the ganglion cell is stronger when the light exposure precedes vestibular stimulus by the optimal inter-stimulus interval (ISI). Since the ganglion cell has no excitatory input synapses, the increased rebound must result from a mechanism internal to the cell that heightens its background activity during pairing at the optimal ISI. The mechanism must be time dependent and must be able to distinguish between the inhibitory signal which comes from the B cell and that which comes from the caudal hair cell. To achieve this result, two mechanisms must interact.

The first mechanism enhances the inhibitory effect of the caudal hair cell on the ganglion cell. This "caudal inhibition enhancer", CIE, is triggered by pulses from the B cell. The CIE has the property that it rises exponentially toward 1.0 when a pulse is seen at the synapse from the B cell and decays toward zero when no such pulses are received.

The second mechanism provides an increase in the background activity of the optic ganglion when the cell is hyperpolarized; it is a fatigue effect at the synapse from the caudal hair cell. This synapse specific fatigue (SSF) rises toward 1.0 as any of the inhibitory synapses onto the ganglion receive a pulse, and decays toward zero when there is no incoming inhibitory pulse. Note that this second order control causes fatigue at the synapse between the caudal hair cell and the ganglion whenever any inhibitory pulse is incident on the ganglion.

Control of the ISI resides in the interaction of these two mechanisms. The efficacy of an inhibitory pulse from the caudal cell upon the ganglion cell is determined by the product of CIE and (1 - SSF), the "ISI convolver." With light exposure alone or when caudal stimulation follows light, the CIE rises toward 1.0 along with the SSF. Initially, (1 - SSF) is close to 1.0 and the CIE term dominates the convolver function. As CIE approaches 1.0, the (1 - SSF) term brings the convolver toward 0. At some intermediate time, the ISI convolver is at a maximum.

When vestibular stimulus precedes light exposure, the SSF rises at the start of the vestibular stimulus while the CIE remains at 0. When light exposure then begins, the CIE rises, but by then (1 - SSF) is approaching zero, and the convolver does not reach any significant value.

The result of this interaction is that when caudal stimulation follows light by the optimal ISI, the inhibition of the ganglion will be maximal. This causes heightened background activity in the ganglion. Upon cessation of stimulus, the heightened background activity will express itself by rapidly depolarizing the ganglion membrane, thereby bringing about the desired rebound firing.

## SHUNTING OF THE PHOTORECEPTOR B CELL DURING EXPOSURE TO LIGHT

In experiments in which light and vestibular stimulus are paired, both the B cell and the caudal hair statocyst cell fire strongly. There is an inhibitory synapse from the hair cell to the B cell (see Figure 1). Without shunting, the hair cell output pulses interfere with the effect of light on the B cell and prevent it from arriving at a level of depolarization necessary for learning. This is contrary to experimental data which shows that the response of the B cell to light (during the light pulse) is constant whether or not vestibular stimulus is present. Biological experiments have determined that while light is on, the B cell shunting resistance is very low making the cell insensitive to incoming pulses.

Figures 5-8 summarize the current performance of the model. Figures 6, 7, and 8 present the response to a light pulse of the untrained, sham trained (unpaired light and turbulence), and trained (paired light and turbulence) model of the four neuron network.

## DISCUSSION

The model developed here is more complex than those generally employed in neural network research because the mechanisms invoked are primarily second order controls. Furthermore, while we operated under a paradigm of minimal commitment (no new features unless needed), the functional requirements of network demanded that differentiating features be added to the cells. The model reproduces the

electrophysiological measure-ments   in Hermissenda that are indicative of associative learning.  These results call into question the notion that linear and quasi-linear summing elements are capable of emulating neural activity and the learning inherent in neural systems.

This preliminary modeling effort has already resulted in a greater understanding of biological systems by 1) modeling experiments which cannot be performed in vivo, 2) testing theoretical constructs on the model, and 3) developing hypotheses and proposing neurophysiological experiments.  The effort has also significantly assisted in the  development of neural network algorithms by uncovering the necessary and sufficient components for learning at the neurophysiological level.

## Acknowledgements

The authors wish to acknowledge the contribution of Peter Tchoryk, Jr. for assistance in performing the simulation experiments and Kim T. Blackwell for many fruitful discussions of the work and the manuscript. This work was supported in part by ONR contract N00014-88-K-0659.

## References

[1] Alkon, D.L. Memory Traces in the Brain, Cambridge University Press, London,. 1987 and publications referenced therein.

[2] Alkon, D.L. Learning in a Marine Snail.   Scientific American, 249: 70-84 (1983).

[3] Alkon, D.L., Sakakibara, M., Forman, M., Harrigan, R., Lederhendler, J., and Farley, J.   Reduction of two voltage-dependent $K^+$ currents mediates retention of learning association.   Behavioral and Neural Biol. 44:278-300 (1985).

[4] Kuffler, S.W., and Nicholas, J.G. From Neural to Brain: A Cellular Approach the the Function of the Nervous System. Seinauer Assoc., Publ., Sunderland, MA (1986).

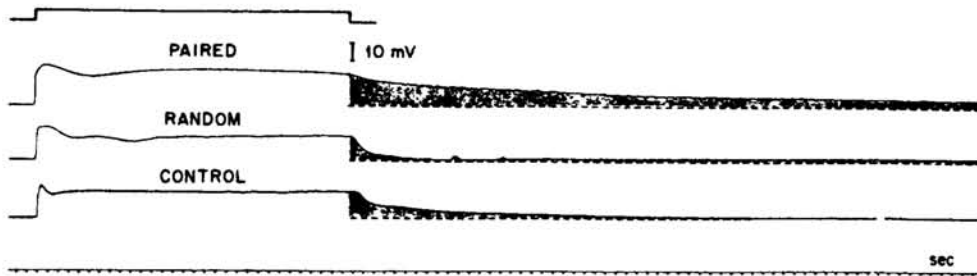

Figure 5.   Prolongation of B cell post-stimulus membrane depolarization consequent to learning (exposure to paired stimuli).
From: West, A. Barnes, E., Alkon, D.L.    Primary changes of voltage responses during retention of associative learning. J. of Neurophysiol. 48:1243-1255 (1982). Note the increase in size of the shaded area which is the effect of learning.

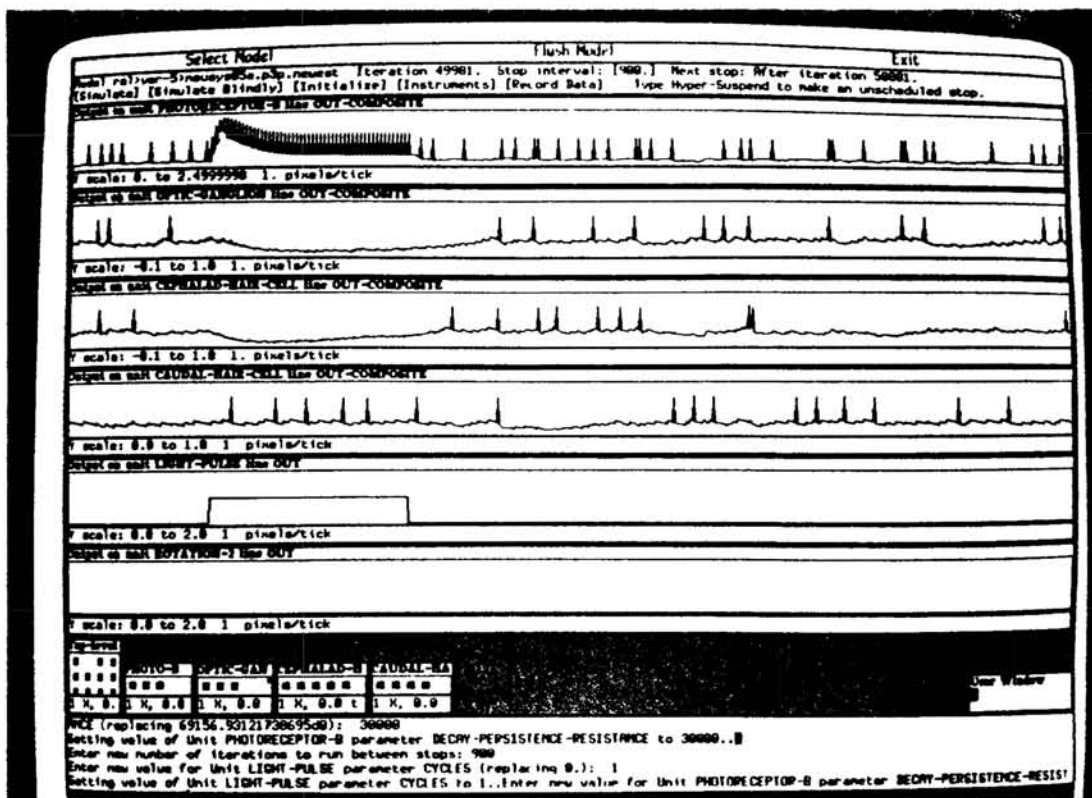

Figure 6.   Naive model: response of untrained ("control" in Fig. 5) model to light.

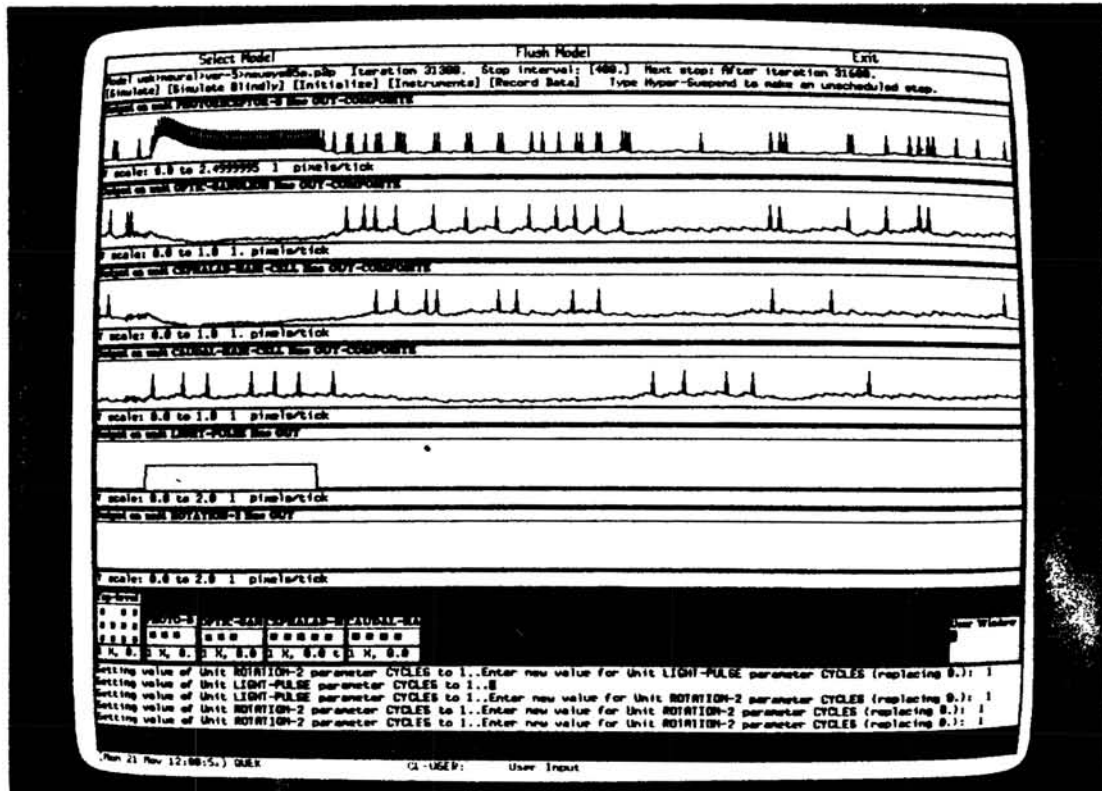

Figure 7. Sham training: response of model to light following presentation of 26 randomly alternating ("-unpaired" in Fig. 5) light and turbulence inputs.

**References (Continued)**

[5]Zipser, D., and Rabin, D.   P3: A Parallel Network Simulating System.   In Parallel Distributed Processing, Vol. I., Chapter 13.  Rumelhart, McClelland, and the PDP Group, Eds.  MIT Press (1986).

[6]Buhmann, J., and Schulten, K.  Influence of Noise on the Function of a "Physiological" Neural Network.  Biological Cybernetics 56:313-328 (1987).

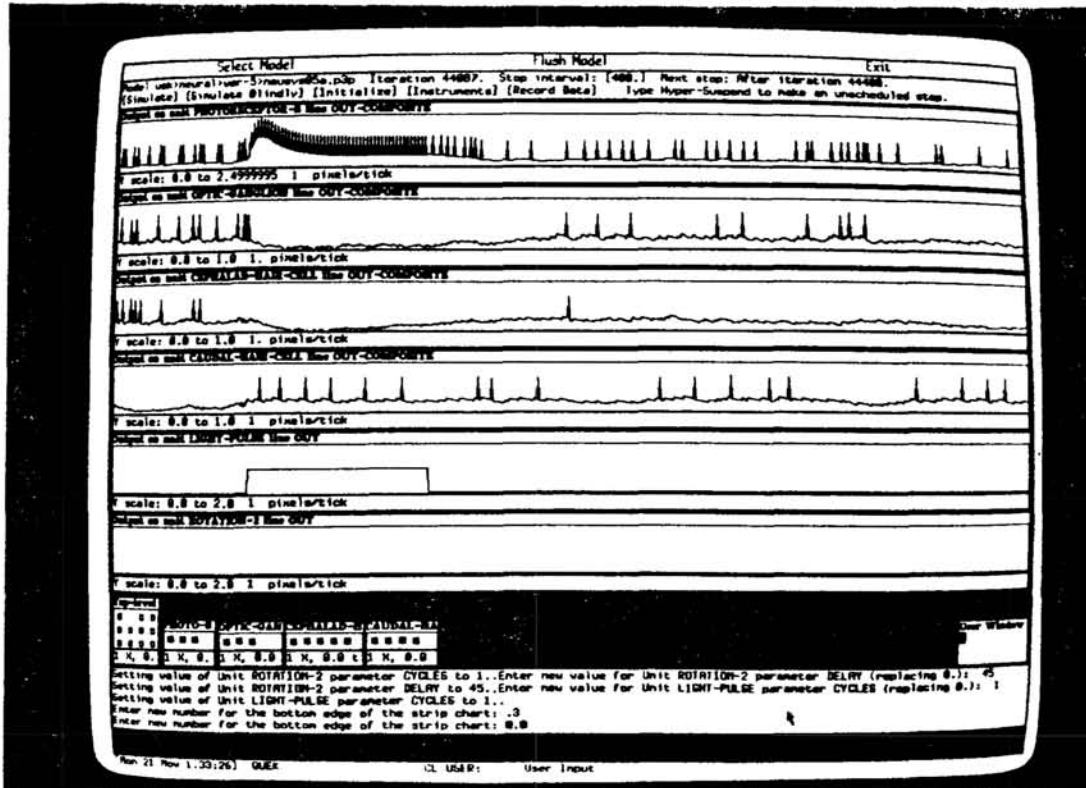

Figure 8. Trained network: response of network to light following presentation of 13 light and turbulence input at optimum ISI. The top trace of this figure is the B cell response to light alone. Note that an increased firing frequency and active membrane potential is maintained after the cessation of light, compared to Figures 6 and 7. This is analogous to what may be seen in Hermissenda, Figure 5. Note also that the optic ganglion and the cephalad hair cell (trace 2 and 3 of this figure) show a decreased post-stimulus firing rate compared with that of Figures 6 and 7.